# Denoising and Dimension Reduction in Feature Space

**Mikio L. Braun**
Fraunhofer Institute[1]
FIRST.IDA
Kekuléstr. 7, 12489 Berlin
mikio@first.fhg.de

**Joachim Buhmann**
Inst. of Computational Science
ETH Zurich
CH-8092 Zürich
jbuhmann@inf.ethz.ch

**Klaus-Robert Müller[2,1]**
Technical University of Berlin[2]
Computer Science
Franklinstr. 28/29, 10587 Berlin
krm@cs.tu-berlin.de

## Abstract

We show that the relevant information about a classification problem in feature space is contained up to negligible error in a finite number of leading kernel PCA components if the kernel matches the underlying learning problem. Thus, kernels not only transform data sets such that good generalization can be achieved even by linear discriminant functions, but this transformation is also performed in a manner which makes economic use of feature space dimensions. In the best case, kernels provide efficient implicit representations of the data to perform classification. Practically, we propose an algorithm which enables us to recover the subspace and dimensionality relevant for good classification. Our algorithm can therefore be applied (1) to analyze the interplay of data set and kernel in a geometric fashion, (2) to help in model selection, and to (3) de-noise in feature space in order to yield better classification results.

## 1 Introduction

Kernel machines use a kernel function as a non-linear mapping of the original data into a high-dimensional feature space; this mapping is often referred to as empirical kernel map [6, 11, 8, 9]. By virtue of the empirical kernel map, the data is ideally transformed such that a linear discriminative function can separate the classes with low generalization error, say via a canonical hyperplane with large margin. The latter is used to provide an appropriate mechanism of capacity control and thus to "protect" against the high dimensionality of the feature space.

The idea of this paper is to add another aspect, not covered by this picture. We will show theoretically that if the learning problem matches the kernel well, the relevant information of a supervised learning data set is always contained in a finite number of leading kernel PCA components (that is, the label information projected to the kernel PCA directions), up to negligible error. This result is based on recent approximation bounds dealing with the eigenvectors of the kernel matrix which show that if a function can be reconstructed using only a few kernel PCA components asymptotically, then the same already holds in a finite sample setting, even for small sample sizes.

Consequently, the use of a kernel function not only greatly increases the expressive power of linear methods by non-linearly transforming the data, but it does so ensuring that the high dimensionality of the feature space will not become overwhelming: the relevant information for classification will stay confined within a comparably *low* dimensional subspace. This finding underlines the efficient use of data that is made by kernel machines using a kernel suited to the problem. While the number of data points stays constant for a given problem, a smart choice of kernel permits to make better use of the available data at a favorable "data point per effective dimension"-ratio, even for infinite-dimensional feature spaces. Furthermore we can use de-noising techniques in feature space, much in the spirit of Mika et al. [8, 5] and thus regularize the learning problem in an elegant manner.

Let us consider an example. Figure 1 shows the first six kernel PCA components for an example data set. Above each plot, the variance of the data along this direction and the contribution of this

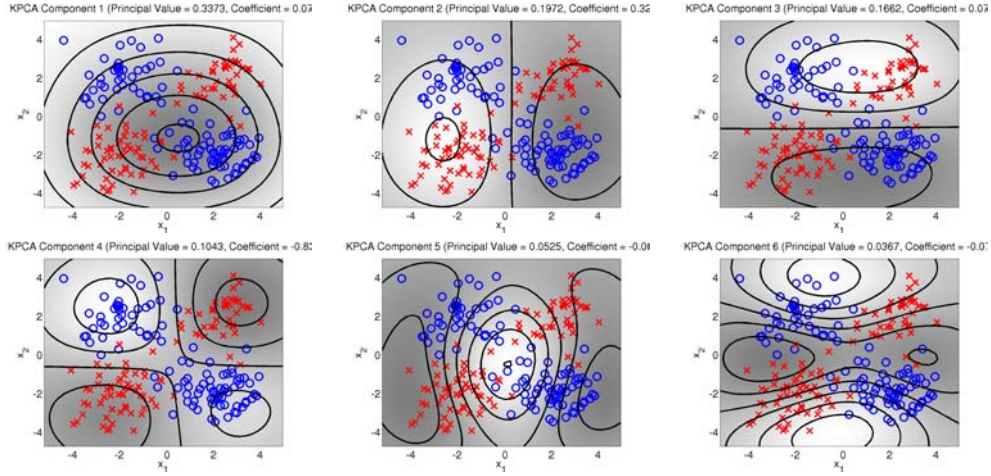

Figure 1: Although the data set is embedded into a high-dimensional manifold, not all directions contain interesting information. Above the first six kernel PCA components are plotted. Of these, only the fourths is highly relevant for the learning problem. Note, however, that this example is atypical in having a single relevant component. In general, several components will have to be combined to construct the decision boundary.

component to the class labels are plotted (normalized such that the maximal possible contribution is one[1]). Of these six components, only the fourth contributes significantly to the class memberships. As we will see below, the contributions in the other directions is mostly noise. This is true especially for components with small variance. Therefore, after removing this noise, a finite number of components suffice to represent the optimal decision boundary.

The dimensionality of the data set in feature space is characteristic for the relation between a data set and a kernel. Roughly speaking, the relevant dimensionality of the data set corresponds to the complexity of the learning problem when viewed through the lens of the kernel function. This notion of complexity relates the number of data points required by the learning problem and the noise, as a small relevant dimensionality enables the de-noising of the data set to obtain an estimate of the true class labels, making the learning process much more stable. This combination of dimension and noise estimate allows us to distinguish among data sets showing weak performance which might either be complex or noisy.

To summarize the main contributions of this paper: (1) We provide theoretical bounds showing that the relevant information (defined in section 2) is actually contained in the leading projected kernel principal components under appropriate conditions. (2) We propose an algorithm which estimates the relevant dimensionality of the data set and permits to analyze the appropriateness of a kernel for the data set, and thus to perform model selection among different kernels. (3) We show how the dimension estimate can be used in conjunction with kernel PCA to perform effective de-noising. We analyze some well-known benchmark data sets and evaluate the performance as a de-noising tool in Section 5. Note that we do not claim to obtain better performance within our framework when compared to, for example, cross-validation techniques. Rather, we are on par. Our contribution is to foster an understanding about a data set and to gain better insights of whether a mediocre classification result is due to intrinsic high dimensionality of the data or overwhelming noise level.

## 2 The Relevant Information and Kernel PCA Components

In this section, we will define the notion of the relevant information contained in the class labels, and show that the location of this vector with respect to the kernel PCA components is linked to the scalar products with the eigenvectors of the kernel matrix.

Let us start to formalize the ideas introduced so far. As usual, we will consider a data set $(X_1, Y_1)$, $\ldots, (X_n, Y_n)$ where the $X$ lie in some space $\mathcal{X}$ and the $Y$ are in $\mathcal{Y} = \{\pm 1\}$. We assume that the $(X_i, Y_i)$ are drawn i.i.d. from $\mathbf{P}_{\mathcal{X} \times \mathcal{Y}}$. In kernel methods, the data is non-linearly mapped into some feature space $\mathcal{F}$ via the feature map $\Phi$. Scalar products in $\mathcal{F}$ can be computed by the kernel $k$ in closed form: $\langle \Phi(x), \Phi(x') \rangle = k(x, x')$. Summarizing all the pairwise scalar products results in the (normalized) kernel matrix $\mathbf{K}$ with entries $k(X_i, X_j)/n$.

We wish to summarize the information contained in the class label vector $Y = (Y_1, \ldots, Y_n)$ about the optimal decision boundary. We define the *relevant information vector* as the vector $G = (\mathbf{E}(Y_1|X_1), \ldots, \mathbf{E}(Y_n|X_n))$ containing the expected class labels for the objects in the training set. The idea is that since $\mathbf{E}(Y|X) = P(Y = 1|X) - P(Y = -1|X)$, the sign of $G$ contains the relevant information on the true class membership by telling us which class is more probable. The observed class label vector can be written as $Y = G - N$ with $N = G - Y$ denoting the noise in the class labels. We want to study the relation of $G$ with respect to the kernel PCA components. The following lemma relates projections of $G$ to the eigenvectors of the kernel matrix $\mathbf{K}$:

**Lemma 1** *The $k$th kernel PCA component $f_k$ evaluated on the $X_i$s is equal to the $k$th eigenvector[2] of the kernel matrix $\mathbf{K}$: $(f_k(X_1), \ldots, f_k(X_n)) = u_k$. Consequently, the projection of a vector $Y \in \mathbb{R}^n$ to the leading $d$ kernel PCA components is given by $\pi_d(Y) = \sum_{k=1}^{d} u_k u_k^\top Y$.*

**Proof** The kernel PCA directions are given as (see [10]) $v_k = \sum_{i=1}^{n} \alpha_i \Phi(X_i)$, where $\alpha_i = [u_k]_i / l_k$, $[u_k]_i$ denoting the $i$th component of $u_k$, and $l_k, u_k$ being the eigenvalues and eigenvectors of the kernel matrix $\mathbf{K}$. Thus, the $k$th PCA component for a point $X_j$ in the training set is

$$f_k(X_j) = \langle \Phi(X_j), v_k \rangle = \frac{1}{l_k} \sum_{i=1}^{n} \langle \Phi(X_j), \Phi(X_i) \rangle [u_k]_i = \frac{1}{l_k} \sum_{i=1}^{n} k(X_j, X_i) [u_k]_i.$$

The sum computes the $j$th component of $\mathbf{K} u_k = l_k u_k$, because $u_k$ is an eigenvector of $\mathbf{K}$. Therefore

$$f_k(X_j) = \frac{1}{l_k} [l_k u_k]_j = [u_k]_j.$$

Since the $u_k$ are orthogonal ($\mathbf{K}$ is a symmetric matrix), the projection of $Y$ to the space spanned by the first $d$ kernel PCA components is given by $\sum_{i=1}^{d} u_i u_i^\top Y$. ∎

## 3 A Bound on the Contribution of Single Kernel PCA Components

As we have just shown, the location of $G$ is characterized by its scalar products with the eigenvectors of the kernel matrix. In this section, we will apply results from [1, 2] which deal with the asymptotic convergence of spectral properties of the kernel matrix to show that the decay rate of the scalar products are linked to the decay rate of the kernel PCA principal values.

It is clear that we cannot expect $G$ to generally locate favorably with respect to the kernel PCA components, but only when there is some kind of match between $G$ and the chosen kernel. This link will be established by asymptotic considerations. Kernel PCA is closely linked to the spectral properties of the kernel matrix, and it is known [3, 4] that the eigenvalues and the projections to eigenspaces converge. Their asymptotic limits are given as the eigenvalues $\lambda_i$ and eigenfunctions $\psi_i$ of the integral operator $T_k f = \int k(\cdot, x) f(x) \mathbf{P}_{\mathcal{X}}(dx)$ defined on $L^2(\mathbf{P}_{\mathcal{X}})$, where $\mathbf{P}_{\mathcal{X}}$ is the marginal measure of $\mathbf{P}_{\mathcal{X} \times \mathcal{Y}}$ which generates our samples. The eigenvalues and eigenfunctions also occur in the well-known Mercer's formula: By Mercer's theorem, $k(x, x') = \sum_{i=1}^{\infty} \lambda_i \psi_i(x) \psi_i(x')$. The asymptotic counterpart of $G$ is given by the function $g(x) = \mathbf{E}(Y|X = x)$.

We will encode fitness between $k$ and $g$ by requiring that $g$ lies in the image of $T_k$. This is equivalent to saying that there exists a sequence $(\alpha_i) \in \ell^2$ such that $g = \sum_{i=1}^{\infty} \lambda_i \alpha_i \psi_i$.[3] Under this condition, the scalar products decay as quickly as the eigenvalues, because $\langle g, \psi_i \rangle = \lambda_i \alpha_i = O(\lambda_i)$. Because of the known convergence of spectral projections, we can expect the same behavior asymptotically

from the finite sample case. However, the convergence speed is the crucial question. This question is not trivial, because eigenvector stability is known to be linked to the gap between the corresponding eigenvalues, which will be fairly small for small eigenvalues. In fact, for example, the results from [14] do not scale properly with the corresponding eigenvalue, such that the bounds are too loose. A number of recent results on the spectral properties of the kernel matrix [1, 2] specifically deal with error bounds for small eigenvalues and their associated spectral projections. Using these results, we obtain the following bound on $u_i^\top G$.[4]

**Theorem 1** *Let $g = \sum_{i=1}^{\infty} \alpha_i \lambda_i \psi_i$ as explained above, and let $G = (g(X_1), \ldots, g(X_n))$. Then, with high probability.*

$$\frac{1}{\sqrt{n}} |u_i^\top G| < 2l_i a_r c_i (1 + O(rn^{-1/4}))$$
$$+ r a_r \Lambda_r O(1) + T_r + \sqrt{A T_r} O(n^{-1/4}) + r a_r \sqrt{\Lambda_r} O(n^{-1/2}),$$

*where $r$ balances the different terms ($1 \le r \le n$), $c_i$ measures the size of the eigenvalue cluster around $l_i$, $a_r = \sum_{i=1}^{r} |\alpha_i|$ is a measure of the size of the first $r$ components, $\Lambda_r$ is the sum of all eigenvalues smaller than $\lambda_r$, $A$ is the supremum norm of $g$, and $T_r$ is the error of projecting $g$ to the space spanned by the first $r$ eigenfunctions.*

The bound consists of a part which scales with $l_i$ (first term) and a part which does not (remaining terms). Typically, the bound initially scales with $l_i$ until the non-scaling part dominates the bound for larger $i$. These two parts are balanced by $r$. However, note that all terms which do not scale with $l_i$ will typically be small: for smooth kernels, the eigenvalues quickly decay to zero as $r \to \infty$. The related quantities $\Lambda_r$, and $T_r$, will also decay to zero at slightly slower rates. Therefore, by adjusting $r$ (as $n \to \infty$), the non-scaling part can be made arbitrarily small, leading to a small bound on $|u_i^\top G|$ for larger $i$.

Put differently, the bound shows that the relevant information vector $G$ (as introduced in Section 2) is contained in a number of leading PCA components up to a negligible error. The number of dimensions depends on the asymptotic coefficients $\alpha_i$ and the decay rate of the asymptotic eigenvalues of $k$. Since this rate is related to the smoothness of the kernel function, the dimension will be small for smooth kernels whose leading eigenfunctions $\psi_i$ permit good approximation of $g$.

## 4   The Relevant Dimension Estimation Algorithm

In this section, we will propose the relevant dimension estimation (RDE) algorithm which estimates the dimensionality of the relevant information from a finite sample, allowing us to analyze the fit between a kernel function and a data set in a practical way.

**Dimension Estimation**   We propose an approach which is motivated by the geometric findings explained above. Since $G$ is not known, we can only observe the contributions of the kernel PCA components to $Y$, which can be written as $Y = G + N$ (see Section 2). The contributions $u_i^\top Y$ will thus be formed as a superposition of $u_i^\top Y = u_i^\top G + u_i^\top N$. Now, by Theorem 1, we know that $G$ will be very close to zero for the latter coefficients, while on the other hand, the noise $N$ will be equally distributed over all coefficients. Therefore, the kernel PCA coefficients $s = u_i^\top Y$ will have the shape of an evenly distributed noise floor $u_i^\top N$ from which the coefficients $u_i^\top G$ of the relevant information protrude (see Figure 2(b) for an example).

We thus propose the following algorithm: Given a fixed kernel $k$, we estimate the true dimension by fitting a two component model to the coordinates of the label vector. Let $s = (u_1^\top Y, \ldots, u_n^\top Y)$. Then, assume that

$$s_i \sim \begin{cases} \mathcal{N}(0, \sigma_1^2) & 1 \le i \le d \\ \mathcal{N}(0, \sigma_2^2) & d < i \le n. \end{cases}$$

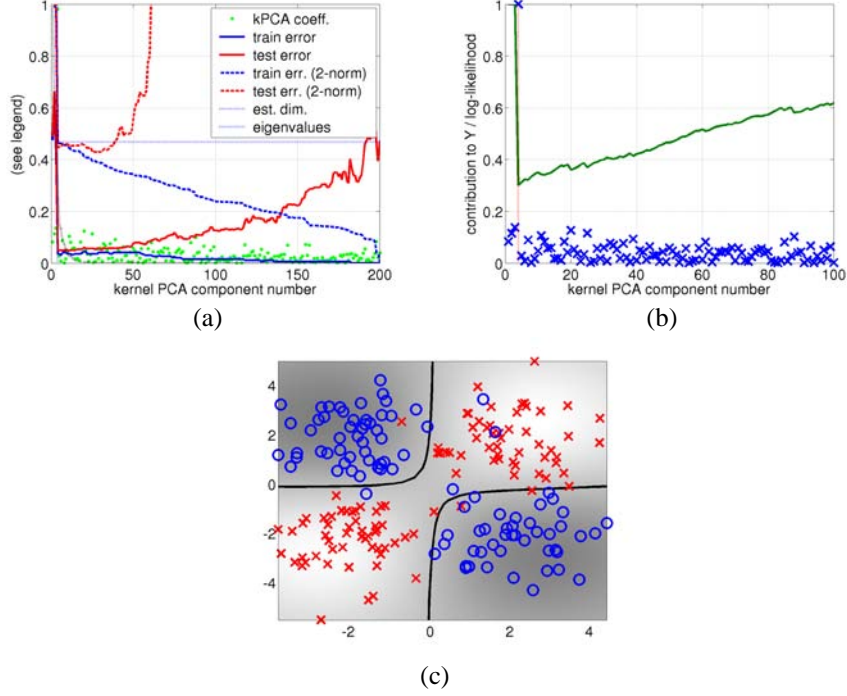

(a)                                                    (b)

(c)

Figure 2: Further plots on the toy example from the introduction. (a) contains the kernel PCA component contributions (dots), and the training and test error by projecting the data set to the given number of leading kernel PCA components. (b) shows the negative log-likelihood of the two component model used to estimate the dimensionality of the data. (c) The resulting fit when using only the first four components.

We select the $d$ minimizing the negative log-likelihood, which is proportional to

$$\ell(d) = \frac{d}{n} \log \sigma_1^2 + \frac{n-d}{n} \log \sigma_2^2, \qquad \text{with} \qquad \sigma_1^2 = \frac{1}{d} \sum_{i=1}^{d} s_i^2, \ \sigma_2^2 = \frac{1}{n-d} \sum_{i=d+1}^{n} s_i^2. \quad (1)$$

**Model Selection for Kernel Choice**  For different kernels, we again use the likelihood and select the kernel which leads to the best fit in terms of the likelihood. If the kernel width does not match the scale of the structure of the data set, the fit of the two component model will be inferior: for very small or very large kernels, the kernel PCA coefficients of $Y$ have no clear structure, such that the likelihood will be small. For example, for Gaussian kernels, for very small kernel widths, noise is interpreted as relevant information, such that there appears to be no noise, only very high-dimensional data. On the other hand, for very large kernel widths, any structure will be indistinguishable from noise such that the problem appears to be very noisy with almost no structure. In both cases, fitting the two component model will not work very well, leading to large values of $\ell$.

**Experimental Error Estimation**  The estimated dimension can be used to estimate the noise level present in the data set. The idea is to measure the error between the projected label vector $\hat{G} = \pi_d(Y)$, which approximates the true label information $G$. The resulting number $\hat{\text{err}} = \frac{1}{n} \sum_{i=1}^{n} 1\{[\hat{G}]_i \neq Y_i\}$ is an estimate of the fraction of misclassified examples in the training set, and therefore an estimate for the noise level in the class labels.

**A Note on Consistency**  Both the estimate of $\hat{G}$ and the noise level are consistent if the estimated dimension $d$ scales sub-linearly with $n$. The argument can be sketched as follows: since the kernel PCA components do not depend on $Y$, the noise $N$ contained in $Y$ is projected to a random subspace of dimension $d$. Therefore, $\frac{1}{n}\|\pi_d(N)\|^2 \approx \frac{d}{n}(\frac{1}{n}\|N\|^2) \rightarrow 0$ as $n \rightarrow \infty$, since $d/n \rightarrow 0$ and $\frac{1}{n}\|N\|^2 \rightarrow \mathbf{E}(N^2)$. Empirically, $d$ was found to be rather stable, but in principle, the condition

on $d$ could even be enforced by adding a small sub-linear term (for example, $\sqrt{n}$, or $\log n$, to the estimated dimension $d$).

## 5   Experiments

**Toy Data Set**   Returning to the toy example from the introduction, let us now take a closer look at this data set. In Figure 2(a), the spectrum for the toy data set is plotted. We can see that every kernel PCA component contributes to the observed class label vector. However, most of these contributions are noise, since the classes are overlapping. The RDE method estimates that only the first four components are relevant. This behavior of the algorithm can also be seen from the training and independent test error measured on a second data set of size 1000 which can also be found in this plot. In Figure 2(b), the log-likelihoods from (1) are shown, and one observes a well pronounced minimum. Finally, in Figure 2(c), the resulting fit is shown.

**Benchmark data sets**   We performed experiments on the classification learning sets from [7]. For each of the data sets, we de-noise the data set using a family of rbf kernels by projecting the class labels to the estimated number of leading kernel PCA components. The kernel width is also selected automatically using the achieved log-likelihood as described above. The width of the rbf kernel is selected from 20 logarithmically spaced points between $10^{-2}$ and $10^4$ for each data set.

For the dimension estimation task, we compare our RDE method to a dimensionality estimate based on cross-validation. More concretely, the matrix $\mathbf{S} = \sum_{i=1}^{d} u_i u_i^\top$ computes the projection to the leading $d$ kernel PCA components. Interpreting the matrix $\mathbf{S}$ as a linear fit matrix, the leave-one-out cross-validation error can be computed in closed form (see [12])[5], since $\mathbf{S}$ is diagonal with respect to the eigenvector basis $u_i$. Evaluating the cross-validation error for all dimensions and for a number of kernel parameters, one can select the best dimension and kernel parameter. Since the cross-validation can be computed efficiently, the computational demands of both methods are equal. Table 3 shows the resulting dimension estimates. We see that both methods perform on par, which shows that the strong structural prior assumption underlying RDE is justified.

For the de-noising task, we have compared a (unregularized) least-squares fit in the reduced feature space (kPCR) against kernel ridge regression (KRR) and support vector machines (SVM) on the same data set. The resulting test errors are plotted also in Table 3. We see that a relatively simple method on the reduced features leads to classification which is on par with the state-of-the-art competitors. Also note that the estimated error rates match the actually observed error rates quite well, although there is a tendency to under-estimate the true error.

Finally, inspecting the estimated dimension and noise level reveals that the data sets *breast-cancer*, *diabetis*, *flare-solar*, *german*, and *titanic* all have only moderately large dimensionalities. This suggest that these data sets are inherently noisy and better results cannot be expected, at least within the family of rbf kernels. On the other hand, the data set *image* seems to be particularly noise free, given that one can achieve a small error in spite of the large dimensionality. Finally, the *splice* data set seems to be a good candidate to benefit from more data.

## 6   Conclusion

Both in theory and on practical data sets, we have shown that the relevant information in a supervised learning scenario is contained in the leading projected kernel PCA components if the kernel matches the learning problem. The theory provides a consistent estimation for the expected class labels and the noise level. This behavior complements the common statistical learning theoretical view on kernel based learning with insight on the interaction of data and kernel: A well chosen kernel (a) makes the model estimate efficiently and generalize well, since only a comparatively low dimensional representation needs to be learned for a fixed given data size and (b) permits a de-noising step that discards some void projected kernel PCA directions and thus provides a regularized model.

Practically, our RDE algorithm automatically selects the appropriate kernel model for the data and extracts as additional side information an estimate of the effective dimension and estimated expected

| data set | dim | dim (cv) | est. error rate | kPCR | KRR | SVM |
|---|---|---|---|---|---|---|
| banana | 24 | 26 | $8.8 \pm 1.5$ | $11.3 \pm 0.7$ | $10.6 \pm 0.5$ | $11.5 \pm 0.7$ |
| breast-cancer | 2 | 2 | $25.6 \pm 2.1$ | $27.0 \pm 4.6$ | $26.5 \pm 4.7$ | $26.0 \pm 4.7$ |
| diabetis | 9 | 9 | $21.5 \pm 1.3$ | $23.6 \pm 1.8$ | $23.2 \pm 1.7$ | $23.5 \pm 1.7$ |
| flare-solar | 10 | 10 | $32.9 \pm 1.2$ | $33.3 \pm 1.8$ | $34.1 \pm 1.8$ | $32.4 \pm 1.8$ |
| german | 12 | 12 | $22.9 \pm 1.1$ | $24.1 \pm 2.1$ | $23.5 \pm 2.2$ | $23.6 \pm 2.1$ |
| heart | 4 | 5 | $15.8 \pm 2.5$ | $16.7 \pm 3.8$ | $16.6 \pm 3.5$ | $16.0 \pm 3.3$ |
| image | 272 | 368 | $1.7 \pm 1.0$ | $4.2 \pm 0.9$ | $2.8 \pm 0.5$ | $3.0 \pm 0.6$ |
| ringnorm | 36 | 37 | $1.9 \pm 0.7$ | $4.4 \pm 1.2$ | $4.7 \pm 0.8$ | $1.7 \pm 0.1$ |
| splice | 92 | 89 | $9.2 \pm 1.3$ | $13.8 \pm 0.9$ | $11.0 \pm 0.6$ | $10.9 \pm 0.6$ |
| thyroid | 17 | 18 | $2.0 \pm 1.0$ | $5.1 \pm 2.1$ | $4.3 \pm 2.3$ | $4.8 \pm 2.2$ |
| titanic | 4 | 6 | $20.8 \pm 3.8$ | $22.9 \pm 1.6$ | $22.5 \pm 1.0$ | $22.4 \pm 1.0$ |
| twonorm | 2 | 2 | $2.3 \pm 0.7$ | $2.4 \pm 0.1$ | $2.8 \pm 0.2$ | $3.0 \pm 0.2$ |
| waveform | 14 | 23 | $8.4 \pm 1.5$ | $10.8 \pm 0.9$ | $9.7 \pm 0.4$ | $9.9 \pm 0.4$ |

Figure 3: Estimated dimensions and error rates for the benchmark data sets from [7]. "dim" shows the medians of the estimated dimensionalities over the resamples. "dim (cv)" shows the same quantity, but this time, the dimensions have been estimated by leave-one-out cross-validation. "est. error rate" is the estimated error rate on the training set by comparing the de-noise class labels to the true class labels. The last three columns show the test error rates of three algorithms: "kPCR" predicts using a simple least-squares hyperplane on the estimated subspace in feature space, "KRR" is kernel ridge regression with parameters estimated using leave-one-out cross-validation, and "SVM" are the original error rates from [7].

error for the learning problem. Compared to common cross-validation techniques one could argue that all we have achieved is to find a similar model as usual at a comparable computing time. However, we would like to emphasize that the side information extracted by our procedure contributes to a better understanding of the learning problem at hand: Is the classification result limited due to intrinsic high dimensional structure or are we facing noise and nuisance dimensions? Simulations show the usefulness of our RDE algorithm.

An interesting future direction lies in combining these results with generalization bounds which are also based on the notion of an effective dimension, this time, however, with respect to some regularized hypothesis class (see, for example, [13]). Linking the effective dimension of the data set with the dimension of a learning algorithm, one could obtain data dependent bounds in a natural way with the potential to be tighter than bounds which are based on the abstract capacity of a hypothesis class.

## Footnotes

[1]Note, however, that these numbers do not simply add up, instead the contribution of $a$ and $b$ is $\sqrt{a^2 + b^2}$.

[2]As usual, the eigenvectors are arranged in descending order by corresponding eigenvalue.

[3]A different condition is that $g$ lies in the RKHS generated by $k$. This amounts to saying that $g$ lies in the image of $T_k^{1/2}$. Therefore, the condition used here is slightly more restrictive.

[4]We have tried to reduce the bound to its most prominent features. For a more detailed explanation of the quantities and the proof, see the appendix. Also, the confidence $\delta$ of the "with high probability" part is hidden in the $O(\cdot)$ notation. We have used the $O(\cdot)$ notation rather deliberately to exhibit the dominant constants.

[5]This applies only to the 2-norm. However, as the performance of 2-norm based methods like kernel ridge regression on classification problems show, the 2-norm is also informative on the classification performance.

# References

[1] ML Braun. *Spectral Properties of the Kernel Matrix and Their Application to Kernel Methods in Machine Learning*. PhD thesis, University of Bonn, 2005. Available electronically at http://hss.ulb.uni-bonn.de/diss_online/math_nat_fak/2005/braun_mikio.

[2] ML Braun. Accurate error bounds for the eigenvalues of the kernel matrix. *Journal of Machine Learning Research*, 2006. To appear.

[3] V Koltchinskii and E Giné. Random matrix approximation of spectra of integral operators. *Bernoulli*, 6(1):113–167, 2000.

[4] VI Koltchinskii. Asymptotics of spectral projections of some random matrices approximating integral operators. *Progress in Probability*, 43:191–227, 1998.

[5] S Mika, B Schölkopf, A Smola, K-R Müller, M Scholz, and Gunnar Räthsch. Kernel PCA and de-noising in feature space. In *Advances in Neural Information Processing Systems 11*. MIT Press, 1999.

[6] K-R Müller, S Mika, G Rätsch, K Tsuda, and B Schölkopf. An introduction to kernel-based learning algorithms. *IEEE Transaction on Neural Networks*, 12(2):181–201, May 2001.

[7] G Rätsch, T Onoda, and K-R Müller. Soft margins for AdaBoost. *Machine Learning*, 42(3):287–320, March 2001.

[8] B Schölkopf, S Mika, CJC Burges, P Knirsch, K-R Müller, G Rätsch, and AJ Smola. Input space vs. feature space in kernel-based methods. *IEEE Transactions on Neural Networks*, 10(5):1000–1017, September 1999.

[9] B Schölkopf and AJ Smola. *Learning with Kernels*. MIT Press, 2001.

[10] B Schölkopf, AJ Smola, and K-R Müller. Nonlinear component analysis as a kernel eigenvalue problem. *Neural Computation*, 10:1299–1319, 1998.

[11] V Vapnik. *Statistical Learning Theory*. Wiley, 1998.

[12] G Wahba. *Spline Models For Observational Data*. Society for Industrial and Applied Mathematics, 1990.

[13] T Zhang. Learning bounds for kernel regression using effective data dimensionality. *Neural Computation*, 17:2077–2098, 2005.

[14] L Zwald and G Blanchard. On the convergence of eigenspaces in kernel principal components analysis. In *Advances in Neural Information Processing Systems (NIPS 2005)*, volume 18, 2006.

## A  Proof of Theorem 1

First, let us collect the definitions concerning kernel functions. Let $k$ be a Mercer kernel with $k(x, x') = \sum_{i=1}^{\infty} \lambda_i \psi_i(x) \psi_i(x')$, and $k(x, x) \leq K < \infty$. The kernel matrix of $k$ for an $n$-sample $X_1, \ldots, X_n$ is $[\mathbf{K}]_{ij} = k(X_i, X_j)/n$. Eigenvalues of $\mathbf{K}$ are $l_i$, and its eigenvectors are $u_i$. The kernel $k$ is approximated by the truncated kernel matrix is $k_r(x, x') = \sum_{i=1}^{r} \lambda_i \psi_i(x) \psi_i(x')$, and its kernel matrix is denoted by $\mathbf{K}_r$, whose eigenvalues are $m_i$. The approximation error is measured by $\mathbf{E}_r = \mathbf{K}_r - \mathbf{K}$. We will measure the amount of clustering $c_i$ of the eigenvalues by the number of eigenvalues of $\mathbf{K}_r$ between $l_i/2$ and $2l_i$. The matrix containing the sample vectors of the first $r$ eigenfunctions $\psi_i$ of $k$ is given by $[\mathbf{\Psi}_r]_{i\ell} = \psi_\ell(X_i)/\sqrt{n}$, $1 \leq i \leq n$, $1 \leq \ell \leq r$. Since the eigenfunctions are orthogonal asymptotically, we can expect that the sample vectors of the eigenfunctions converge to either 0 or 1. The error is measured by the matrix $\mathbf{C}_r = \mathbf{\Psi}_r^\top \mathbf{\Psi}_r - \mathbf{I}$. Finally, let $\Lambda_r = \sum_{i=r+1}^{\infty} \lambda_i$.

Next, we collect definitions concerning some function $f$. Let $f = \sum_{i=1}^{\infty} \lambda_i \alpha_i \psi_i$ with $(\alpha_i) \in \ell^2$, and $|f| \leq A < \infty$. The size of the contribution of the first $r$ terms is measured by $a_r = \sum_{i=1}^{r} |\alpha_i|$. Define the error of truncating $f$ to the first $r$ elements of its series expansion by $T_r = (\sum_{i=r+1}^{\infty} \lambda_i^2 \alpha_i^2)^{1/2}$.

The **proof of Theorem 1** is based on performing rough estimates of the bound from Theorem 4.92 in [1]. The bound is

$$\frac{1}{\sqrt{n}} |u_i^\top f(\mathbf{X})| < \min_{1 \leq r \leq n} \left[ l_i D(r, n) + E(r, n) + T(r, n) \right]$$

where the three terms are given by

$$D(r, n) = 2 a_r \|\mathbf{\Psi}_r^+\| c_i, \qquad E(r, n) = 2 r a_r \|\mathbf{\Psi}_r^+\| \|\mathbf{E}^r\|, \qquad T(r, n) = T_r + \sqrt{F T_r} \sqrt[4]{\frac{1}{n\delta}},$$

It holds that $\|\mathbf{\Psi}_r^+\| \leq (1 - \|\mathbf{\Psi}_r^\top \mathbf{\Psi}_r - \mathbf{I}\|)^{1/2} = (1 - \|\mathbf{C}_r\|)^{-1/2}$ ([1], Lemma 4.44). Furthermore, $\|\mathbf{C}_r\| \to 1$ as $n \to \infty$ for fixed $r$. For kernel with bounded diagonal, it holds that with probability larger than $1 - \delta$ ([1], Lemma 3.135) that $\|\mathbf{C}_r\| \leq r \sqrt{r(r+1)K/\lambda_r n \delta} = r^2 O(n^{-1/2})$ with a rather large constant, especially, if $\lambda_r$ is small. Consequently, $\|\mathbf{\Psi}_r^+\| \leq (1 - \|\mathbf{C}_r\|)^{-1/2} = 1 + O(rn^{-1/4})$.

Now, Lemma 3.135 in [1] bounds $\|\mathbf{E}_r\|$ from which we can derive the asymptotic

$$\|\mathbf{E}_r\| \leq \lambda_r + \Lambda_r + \sqrt{\frac{2K\Lambda_r}{n\delta}} = \Lambda_r + \sqrt{\Lambda_r} O(n^{-\frac{1}{2}}),$$

assuming that $K$ will be reasonably small (for example, for rbf-kernels, $K = 1$). Combining this with our rate for $\|\mathbf{\Psi}_r^+\|$, we obtain

$$E(r, n) = 2 r a_r \left( \Lambda_r + \sqrt{\Lambda_r} O(n^{-\frac{1}{2}}) \right) (1 + O(rn^{-\frac{1}{4}})) = 2 r a_r \Lambda_r (1 + O(rn^{-\frac{1}{4}})) + r a_r \sqrt{\Lambda_r} O(n^{-\frac{1}{2}}).$$

Finally, we obtain

$$\frac{1}{\sqrt{n}} |u_i^\top f(\mathbf{X})| = 2 l_i a_r c_i (1 + O(rn^{-\frac{1}{4}}))$$

$$+ 2 r a_r \Lambda_r (1 + O(rn^{-\frac{1}{4}})) + r a_r \sqrt{\Lambda_r} O(n^{-\frac{1}{2}}). + T_r + \sqrt{A T_r} O(n^{-\frac{1}{2}}).$$

If we assume that $\Lambda_r$ will be rather small, we replace $1 + O(rn^{-\frac{1}{4}})$ by $O(1)$ for the second term and obtain the claimed rate. ∎